# Robust Recognition of Noisy and Superimposed Patterns via Selective Attention

**Soo-Young Lee**
Brain Science Research Center
Korea Advanced Institute of Science & Technology
Yusong-gu, Taejon 305-701 Korea
sylee@ee.kaist.ac.kr

**Michael C. Mozer**
Department of Computer Science
University of Colorado at Boulder
Boulder, CO 80309 USA
mozer@cs.colorado.edu

## Abstract

In many classification tasks, recognition accuracy is low because input patterns are corrupted by noise or are spatially or temporally overlapping. We propose an approach to overcoming these limitations based on a model of human selective attention. The model, an early selection filter guided by top-down attentional control, entertains each candidate output class in sequence and adjusts attentional gain coefficients in order to produce a strong response for that class. The chosen class is then the one that obtains the strongest response with the least modulation of attention. We present simulation results on classification of corrupted and superimposed handwritten digit patterns, showing a significant improvement in recognition rates. The algorithm has also been applied in the domain of speech recognition, with comparable results.

## 1 Introduction

In many classification tasks, recognition accuracy is low because input patterns are corrupted by noise or are spatially or temporally overlapping. Approaches have been proposed to make classifiers more robust to such perturbations, e.g., by requiring classifiers to have low input-to-output mapping sensitivity [1]. We propose an approach that is based on human selective attention. People use selective attention to focus on critical features of a stimulus and to suppress irrelevant features. It seems natural to incorporate a selective-attention mechanism into pattern recognition systems for noisy real world applications.

Psychologists have for many years studied the mechanisms of selective attention (e.g., [2]-[4]). However, controversy still exists among competing theories, and only a few models are sufficiently well defined to apply to engineering pattern recognition problems.

Fukushima [5] has incorporated selective attention and attention-switching algorithms into his Neocognitron model, and has demonstrated good recognition performance on superimposed digits. However, the Neocognitron model has many unknown parameters which must be determined heuristically, and its performance is sensitive to the parameter values. Also, its computational requirements are prohibitively expensive for many real-time applications. Rao [6] has also recently introduced a selective attention model based

on Kalman filters and demonstrated classifications of superimposed patterns. However, his model is based on linear systems, and a nonlinear extension is not straightforward. There being no definitive approach to incorporating selective attention into pattern recognition, we propose a novel approach and show it can improve recognition accuracy.

## 2  Psychological Views of Selective Attention

The modern study of selective attention began with Broadbent [7]. Broadbent presented two auditory channels to subjects, one to each ear, and asked subjects to shadow one channel. He observed that although subjects could not recall most of what took place in the unshadowed channel, they could often recall the last few seconds of input on that channel. Therefore, he suggested that the brain briefly stores incoming stimuli but the stimulus information fades and is neither admitted to the conscious mind nor is encoded in a way that would permit later recollection, unless attention is directed toward it. This view is known as an *early filtering* or *early selection* model. Treisman [8] proposed a modification to this view in which the filter merely attenuates the input rather than absolutely preventing further analysis. Although *late-selection* and hybrid views of attention have been proposed, it is clear that early selection plays a significant role in human information processing [3].

The question about where attention acts in the stream of processing is independent of another important issue: what factors drive attention to select one ear or one location instead of another. Attention may be directed based on low-level stimulus features, such as the amplitude of a sound or the color of a visual stimulus. This type of attentional control is often called *bottom up*. Attention may also be directed based on expectations and object knowledge, e.g., to a location where critical task-relevant information is expected. This type of attentional control is often called *top down*.

## 3  A Multilayer Perceptron Architecture for Selective Attention

We borrow the notion of an early selection filter with top-down control and integrate it into a multilayer perceptron (MLP) classifier, as depicted in Figure 1. The dotted box is a standard MLP classifier, and an attention layer with one-to-one connectivity is added in front of the input layer. Although we have depicted an MLP with a single hidden layer, our approach is applicable to general MLP architectures. The $k$th element of the input vector, denoted $x_k$, is gated to the $k$th input of the MLP by an attention gain or filtering coefficient $a_k$. Previously, the first author has shown a benefit of treating the $a_k$'s like ordinary adaptive parameters during training [9]-[12].

In the present work, we fix the attention gains at 1 during training, causing the architecture to behave as an ordinary MLP. However, we allow the gains to be adjusted during classification of test patterns. Our basic conjecture is that recognition accuracy may be improved if attention can suppress noise along irrelevant dimensions and enhance a weak signal along relevant dimensions. "Relevant" and "irrelevant" are determined by top-down control of attention. Essentially, we use knowledge in the trained MLP to determine which input dimensions are critical for classifying a test pattern. To be concrete, consider an MLP trained to classify handwritten digits. When a test pattern is presented, we can adjust the attentional gains via gradient descent so as to make the input as good an example of the class "0" as possible. We do this for each of the different output classes, "0" through "9", and choose the class for which the strongest response is obtained with the smallest

attentional modulation (the exact quantitative rule is presented below). The conjecture is that if the net can achieve a strong response for a class by making a small attentional modulation, that class is more likely to be correct than whichever class would have been selected without applying selective attention.

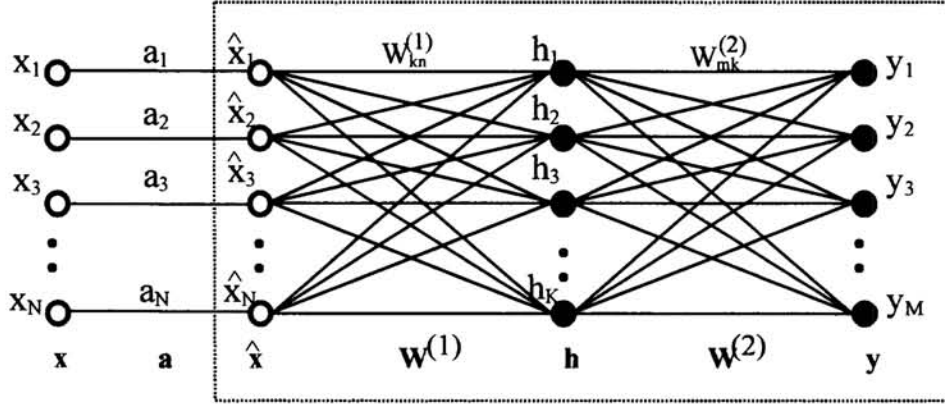

Figure 1: MLP architecture for selective attention

The process of adjusting the attentional gains to achieve a strong response from a particular class—call it the *attention class*—proceeds as follows. First, a target output vector $\mathbf{t}^s = [t^s_1\ t^s_2 \cdots t^s_M]^T$ is defined. For bipolar binary output representations, $t^s_i = 1$ is for the attention class and -1 for the others. Second, the attention gain $a_k$'s are set to 1. Third, the attention gain $a_k$'s are adapted to minimize error $E^s \equiv \frac{1}{2}\sum_i (t^s_i - y_i)^2$ with the given input $\mathbf{x} = [x_1\ x_2 \cdots x_N]^T$ and pre-trained and frozen synaptic weights $W$. The update rule is based on a gradient-descent algorithm with error back-propagation. At the $(n+1)$'th iterative epoch, the attention gain $a_k$ is updated as

$$a_k[n+1] = a_k[n] - \eta(\partial E / \partial a_k)[n] = a_k[n] + \eta\ x_k \delta_k^{(0)}[n] \qquad (1a)$$

$$\delta_k^{(0)} = \sum_j W_{jk}^{(1)} \delta_j^{(1)} \qquad (1b)$$

where $E$ denotes the attention output error, $\delta_j^{(1)}$ the $j$'th attribute of the back-propagated error at the first hidden-layer, and $W_{jk}^{(1)}$ the synaptic weight between the input $\hat{x}_k$ and the $j$'th neuron at the first hidden layer. Finally, $\eta$ is a step size. The attention gains are thresholded to lie in [0, 1]. The application of selective attention to a test example is summarized as follows:

Step 1: Apply a test input pattern to the trained MLP and compute output values.
Step 2: For each of the classes with top $m$ activation values,
&emsp;&emsp;(1) Initialize all attention gain $a_k$'s to 1 and set the target vector $\mathbf{t}^s$.
&emsp;&emsp;(2) Apply the test pattern and attention gains to network and compute output.
&emsp;&emsp;(3) Apply the selective attention algorithm in Eqs.(1) to adapt the attention gains.
&emsp;&emsp;(4) Repeat steps (2) and (3) until the attention process converges.
&emsp;&emsp;(5) Compute an attention measure $M$ on the asymptotic network state.

Step 3: Select the class with a minimum attention measure $M$ as the recognized class.

The attention measure is defined as

$$M \equiv D_I E_O, \tag{2a}$$

$$\begin{aligned} D_I &\equiv \sum_k (x_k - \hat{x}_k)^2 / 2N \\ &= \sum_k x_k^2 (1 - a_k)^2 / 2N \end{aligned}, \tag{2b}$$

$$E_O \equiv \sum_i [t_i - y_i(\hat{\mathbf{x}})]^2 / 2M, \tag{2c}$$

where $D_I$ is the square of Euclidean distance between two input patterns before and after the application of selective attention and $E_O$ is the output error after the application of selective attention. Here, $D_I$ and $E_O$ are normalized with the number of input pixels and number of output classes, respectively. The superscript $s$ for attention classes is omitted for simplicity. To make the measure $M$ a dimensionless quantity, one may normalize the $D_I$ and $E_O$ with the input energy ($\sum_k x_k^2$) and the training output error, respectively. However, it does not affect the selection process in Step 3.

One can think of the attended input $\hat{\mathbf{x}}$ as the minimal deformation of the test input needed to trigger the attended class, and therefore the Euclidean distance between $\mathbf{x}$ and $\hat{\mathbf{x}}$ is a good measure for the classification confidence. In fact, $D_I$ is basically the same quantity minimized by Rao [6]. However, the MLP classifier in our model is capable of nonlinear mapping between the input and output patterns. A nearest-neighbor classifier, with the training data as examples, could also be used to find the minimum-distance class. Our model with the MLP classifier computes a similar function without the large memory and computational requirements.

The proposed selective attention algorithm was tested on recognition of noisy numeral patterns. The numeral database consists of samples of the handwritten digits (0 through 9) collected from 48 people, for a total of 480 samples. Each digit is encoded as a 16x16 binary pixel array. Roughly 16% of the pixels are black and coded as 1; white pixels are coded as 0. Four experiments were conducted with different training sets of 280 training patterns each. A one hidden-layer MLP was trained by back propagation. The numbers of input, hidden, and output neurons were 256, 30, and 10, respectively. Three noisy test patterns were generated from each training pattern by randomly flipping each pixel value with a probability $P_f$, and the 840 test patterns were presented to the network for classification.

In Figure 2, the false recognition rate is plotted as a function of the number of candidates considered for the attentional manipulation, $m$. (Note that the run time of the algorithm is proportional to $m$, but that increasing $m$ does not imply a more lax classification criterion, or additional external knowledge playing into the classification.) Results are shown for three different pixel inversion probabilities, $P_f$ =0.05, 0.1, and 0.15. Considering the average 16% of black pixels in the data, the noisy input patterns with $P_f = 0.15$ correspond to a SNR of approximately 0 dB. For each condition in the figure, the false recognition rates for the four different training sets are marked with an 'o', and the means are connected by the solid curve.

A standard MLP classifier corresponds to $m = 1$ (i.e., only the most active output of the MLP is considered as a candidate response). The false recognition rate is clearly lower when the attentional manipulation is used to select a response from the MLP ($m > 1$). It appears that performance does not improve further by considering more than the top three candidates.

## 4 Attention Switching for Superimposed Patterns

Suppose that we superimpose the binary input patterns for two different handwritten digits using the logical OR operator (the pixels corresponding to the black ink have logical value 1). Can we use attention to recognize the two patterns in sequence? This is an extreme case of a situation that is common in visual pattern recognition—where two patterns are spatially overlapping.

We explore the following algorithm. First, one pattern is recognized with the selective attention process used in Section 3. Second, attention is *switched* from the recognized pattern to the remaining pixels in the image. Switching is accomplished by removing attention from the pixels of the recognized pattern: the attentional gain of an input is clamped to 0 following switching if and only if its value after the first-stage selective attention process was 1 (i.e., that input was attended during the recognition of the first pattern); all other gains are set to 1. Third, the recognition process with selective attention is performed again to recognize the second pattern.

The proposed selective attention and attention switching algorithm was tested for recognition of 2 superimposed numeral data. Again, four experiments were conducted with

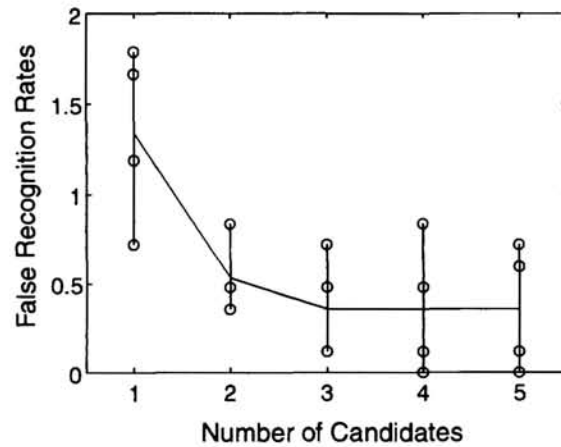

(a) $P_f=0.05$,

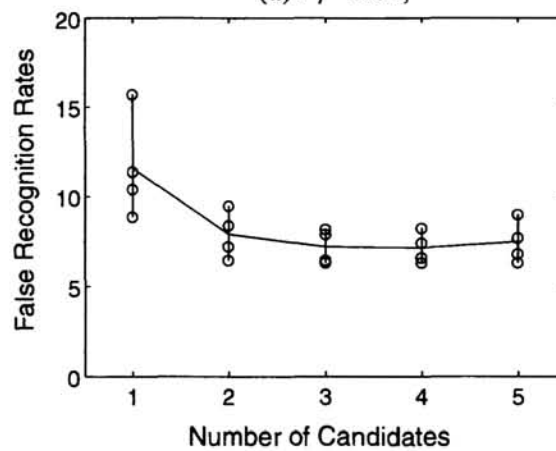

(b) $P_f=0.10$,

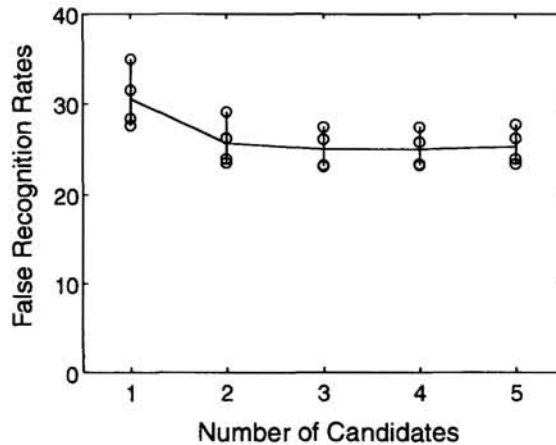

(c) $P_f=0.15$,

Figure 2: False recognition rates for noisy patterns as a function of the number of top candidates. Each binary pixel of training patterns is randomly inverted with a probability $P_f$.

different training sets. For each experiment, 40 patterns were selected from 280 training patterns, and 720 test patterns were generated by superimposing pairs of patterns from different output classes. The test patterns were still binary.

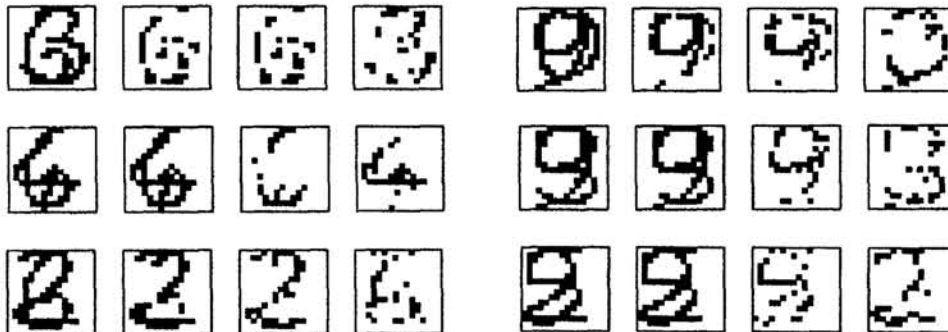

Figure 3: Examples of Selective Attention and Attention Switching

Figure 3 shows six examples of the selective attention and attention switching algorithm in action, each consisting of four panels in a horizontal sequence. The six examples were formed by superimposing instances of the following digit pairs: (6,3), (9,0), (6,4), (9,3), (2,6), and (5,2). The first panel for each example shows the superimposed pattern. The second panel shows the attended input $\hat{x}$ for the first round classification; because this input has continuous values, we have thresholded the values at 0.5 to facilitate viewing in the figure. The third panel shows the masking pattern for attention switching, generated by thresholding the input pattern at 1.0. The fourth panel shows the residual input pattern for the second round classification. The attended input $\hat{x}$ has analog values, but thresholded by 0.5 to be shown in the second rectangles. Figure 3 shows that attention switching is done effectively, and the remaining input patterns to the second classifier are quite visible.

We compared performance for three different methods. First, we simply selected the two MLP outputs with highest activity; this method utilizes neither selective attention. Second, we performed attention switching but did not apply selective attention (i.e., $m=1$). Third, we performed both attention switching and selective attention (with $m=3$). Table 1 summarizes the recognition rates for the first and the second patterns read out of the MLP for the three methods. As hypothesized, attention switching increases the recognition rate for the second pattern, and selective attention increases the recognition rate for both the first and the second pattern.

Table 1: Recognition Rates (%) of Two Superimposed Numeral Patterns

|                                    | First Pattern | Second Pattern |
|------------------------------------|---------------|----------------|
| No selective attention or switching | 91.3          | 62.7           |
| Switching only                     | 91.3          | 75.4           |
| Switching & selective attention    | 95.9          | 77.4           |

# 5 Conclusion

In this paper, we demonstrated a selective-attention algorithm for noisy and superimposed patterns that obtains improved recognition rates. We also proposed a simple attention switching algorithm that utilizes the selective-attention framework to further improve performance on superimposed patterns. The algorithms are simple and easily implemented in feedforward MLPs. Although our experiments are preliminary, they suggest that attention-based algorithms will be useful for extracting and recognizing multiple patterns in a complex background. We have conducted further simulation studies supporting this conjecture in the domain of speech recognition, which we will integrate into this presentation if it is accepted at NIPS.

## Acknowledgements

S.Y. Lee acknowledges supports from the Korean Ministry of Science and Technology. We thank Dr. Y. Le Cun for providing the handwritten digit database.

## References

[1] Jeong D.G., and Lee, S.Y. (1996). Merging backpropagation and Hebbian learning rules for robust classification, *Neural Networks*, 9:1213-1222.

[2] Cowan, N. (1997). *Attention and Memory: An Integrated Framework*, Oxford Univ. Press.

[3] Pashler, H.E. (1998). *The Psychology of Attention*, MIT Press.

[4] Parasuraman, R. (ed.) (1998). *The Attentive Brain*, MIT Press.

[5] Fukushima, K. (1987). Neural network model for selective attention in visual pattern recognition and associative recall, *Applied Optics*, 26:4985-4992.

[6] Rao, R.P.N. (1998). Correlates of attention in a model of dynamic visual recognition. In *Neural Information Processing Systems 10*, MIT Press.

[7] Broadbent, D.E. (1958). *Perception and Communication*. Pergamon Press.

[8] Treisman, A. (1960). Contextual cues in selective listening, *Quarterly Journal of Experimental Psychology*, 12:242-248.

[9] Lee, H.J., Lee, S.Y. Lee, Shin, S.Y., and Koh, B.Y. (1991). TAG: A neural network model for large-scale optical implementation, *Neural Computation*, 3:135-143.

[10]Lee, S.Y., Jang, J.S., Shin, S.Y., & Shim, C.S. (1988). Optical Implementation of Associative Memory with Controlled Bit Significance, *Applied Optics*, 27:1921-1923.

[11]Kruschke, J.K. (1992). ALCOVE: An Examplar-Based Connectionist Model of Category Learning, *Psychological Review*, 99:22-44.

[12]Lee, S.Y., Kim, D.S., Ahn, K.H., Jeong, J.H., Kim, H., Park, S.Y., Kim, L.Y., Lee, J.S., & Lee, H.Y. (1997). Voice Command II: a DSP implementation of robust speech recognition in real-world noisy environments, *International Conference on Neural Information Processing*, pp. 1051-1054, Dunedin, New Zealand.
